# Higher-order Statistical Properties Arising from the Non-stationarity of Natural Signals

**Lucas Parra, Clay Spence**
Adaptive Signal and Image Processing, Sarnoff Corporation
*{lparra,cspence}@sarnoff.com*

**Paul Sajda**
Department of Biomedical Engineering, Columbia University
*ps629@columbia.edu*

## Abstract

We present evidence that several higher-order statistical properties of natural images and signals can be explained by a stochastic model which simply varies scale of an otherwise stationary Gaussian process. We discuss two interesting consequences. The first is that a variety of natural signals can be related through a common model of spherically invariant random processes, which have the attractive property that the joint densities can be constructed from the one dimensional marginal. The second is that in some cases the non-stationarity assumption and only second order methods can be explicitly exploited to find a linear basis that is equivalent to independent components obtained with higher-order methods. This is demonstrated on spectro-temporal components of speech.

## 1 Introduction

Recently, considerable attention has been paid to understanding and modeling the non-Gaussian or "higher-order" properties of natural signals, particularly images. Several non-Gaussian properties have been identified and studied. For example, marginal densities of features have been shown to have high kurtosis or "heavy tails", indicating a non-Gaussian, sparse representation. Another example is the "bow-tie" shape of conditional distributions of neighboring features, indicating dependence of variances [11]. These non-Gaussian properties have motivated a number of image and signal processing algorithms that attempt to exploit higher-order statistics of the signals, e.g., for blind source separation. In this paper we show that these previously observed higher-order phenomena are ubiquitous and can be accounted for by a model which simply varies the scale of an otherwise stationary Gaussian process. This enables us to relate a variety of natural signals to one another and to spherically invariant random processes, which are well-known in the signal processing literature [6, 3]. We present analyses of several kinds of data

from this perspective, including images, speech, magnetoencephalography (MEG) activity, and socio-economic data (e.g., stock market data). Finally we present the results of experiments with algorithms for finding a linear basis equivalent to independent components that exploit non-stationarity so as to require only 2nd-order statistics. This simplification is possible whenever linearity and non-stationarity of independent sources is guaranteed such as for the powers of acoustic signals.

## 2   Scale non-stationarity and high kurtosis

Natural signals can be non-stationary in various ways, e.g. varying powers, changing correlation of neighboring samples, or even non-stationary higher moments. We will concentrate on the simplest possible variation and show in the following sections how it can give rise to many higher-order properties observed in natural signals. We assume that at any given instance a signal is specified by a probability density function with zero mean and unknown scale or power. The signal is assumed non-stationary in the sense that its power varies from one time instance to the next. [1] We can think of this as a stochastic process with samples $z(t)$ drawn from a zero mean distribution $p_z(z)$ with samples possibly correlated in time. We observe a scaled version of this process with time varying scales $s(t) > 0$ sampled from $p_s(s)$,

$$x(t) = s(t)z(t),  \qquad (1)$$

The observable process $x(t)$ is distributed according to

$$p_x(x) = \int_0^\infty ds\, p_s(s)\, p_x(x|s) = \int_0^\infty ds\, p_s(s)\, s^{-1} p_z(\frac{x}{s}). \qquad (2)$$

We refer to $p_x(x)$ as the long-term distribution and $p_z(z)$ as the instantaneous distribution. In essence $p_x(x)$ is a mixture distribution with infinitely many kernels $s^{-1}p_z(\frac{x}{s})$. We would like to relate the sparseness of $p_z(z)$, as measured by the kurtosis, to the sparseness of the observable distribution $p_x(x)$.

Kurtosis is defined as the ratio between the fourth and second cumulant of a distribution [7]. As such it measures the length of the distribution's tails, or the sharpness of its mode. For a zero mean random variable $x$ this reduces up to a constant to

$$K[x] = \frac{\langle x^4 \rangle_x}{\langle x^2 \rangle_x^2} \ , \text{ with } \langle f(x) \rangle_x = \int dx f(x) p_x(x). \qquad (3)$$

In this case we find that the kurtosis of the long-term distribution is always larger than the kurtosis of the instantaneous distribution unless the scale is stationary ([9] and [1] for symmetric $p_z(z)$),

$$K[x] \geq K[z]. \qquad (4)$$

To see this note that the independence of $s$ and $z$ implies, $\langle x^n \rangle_x = \langle s^n \rangle_s \langle z^n \rangle_z$, and therefore, $K[x] = K[z] \langle s^4 \rangle_s / \langle s^2 \rangle_s^2$. From the inequality, $\langle (s^2 - c^2)^2 \rangle_s \geq 0$, which hold for any arbitrary constant $c > 0$, it is easy to show that $\langle s^4 \rangle_s \geq \langle s^2 \rangle_s^2$, where the equality holds for $p_s(s) = \delta(s - c)$. Together this leads to inequality (4), which states that for a fixed scale $s(t)$, i.e. the magnitude of the signal is stationary, the kurtosis will be minimal. Conversely, non-stationary signals, defined as a variable scaling of an otherwise stationary process, will have increased kurtosis.

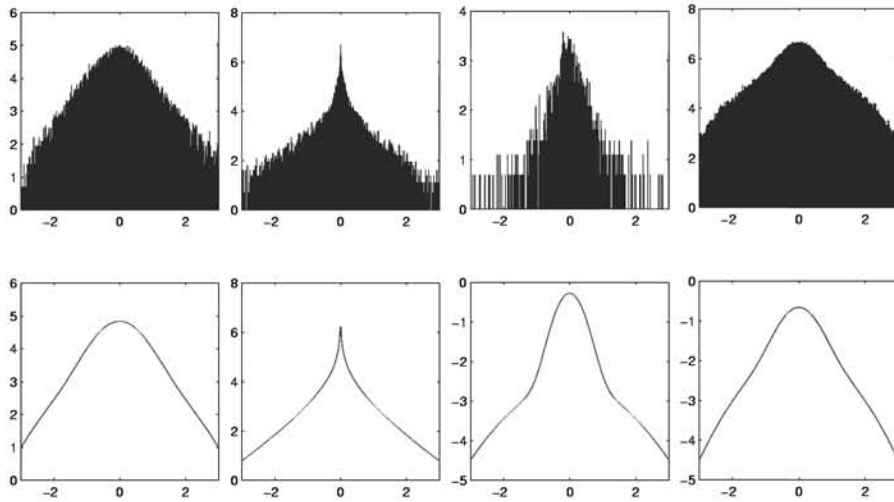

Figure 1: Marginal distributions within 3 standard deviations are shown on a logarithmic scale; left to right: natural image features, speech sound intensities, stock market variation, MEG alpha activity. The measured kurtosis is 4.5, 16.0, 12.9, and 5.3 respectively. On top the empirical histograms are presented and on bottom the model distributions. The speech data has been fit with a Meijer-G function $G_{02}^{20}$ [3]. For the MEG activity, the stock market data and the image features a mixture of zero mean Gaussians was used.

Figure 1 shows empirical plots of the marginal distributions for four natural signals; image, speech, stock market, and MEG data. As image feature we used a wavelet component for a 162x162 natural texture image of sand (presented in [4]). Self-inverting wavelets with a down-sampling factor of three where used. The speech signal is a 2.3 s recording of a female speaker sampled at 8 kHz with a noise level less than -25 dB. The signal has been band limited between 300 Hz and 3.4 kHz corresponding to telephone speech. The market data are the daily closing values of the NY Stock exchange composite index from 02/01/1990 to 04/28/2000. We analyzed the variation from the one day linear prediction value to remove the upwards trend of the last decade. The MEG data is band-passed (10-12 Hz) alpha activity of a in-dependent component of 122 MEG signals. This independendt component exhibits alpha de-synchronization for a visio-motor integration task [10]. One can see that in all four cases the kurtosis is high relative to a Gaussian ($K = 3$). Our claim is that for natural signals, high kurtosis is a natural result of the scale non-stationarity of the signal. Additional evidence comes from the behavior seen in the conditional histograms of the joint distributions, presented in the next section.

## 3 Higher-order properties of joint densities

It has been observed in images that the conditional histograms of joint densities from neighboring features (neighboring in scale, space, and/or orientation) exhibit variance dependencies that cannot be accounted for by simple second-order models [11]. Figure 2 shows empirical conditional histograms for the four types of natural signals we considered earlier. One can see that speech and stock-market data exhibit the same variance dependency or "bow-tie" shape exhibited by images.

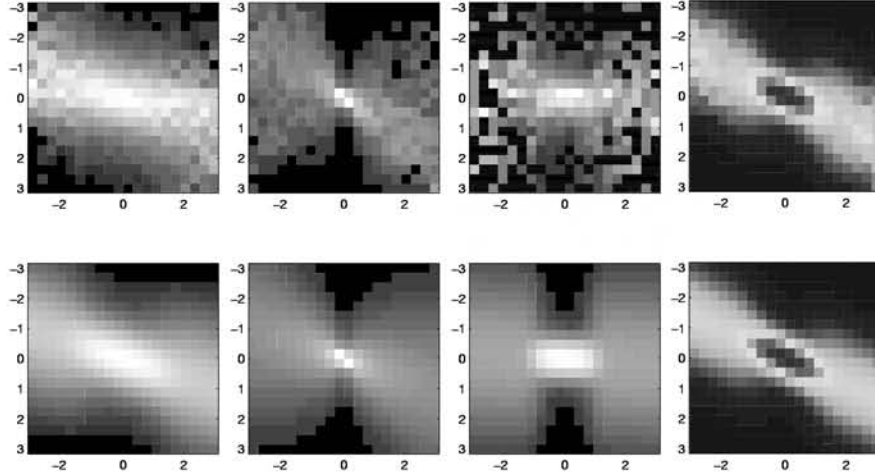

Figure 2: (Top) Empirical conditional histograms and (bottom) model conditional density derived from the one dimensional marginals presented in the previous figure assuming the data is sampled form a SIRP. Good correspondence validates the SIRP assumption which is equivalent to our non-stationary scale model for slow varying scales.

The model of Equation 1 can easily account for this observation if we assume slowly changing scales $s(t)$. A possible explanation is that neighboring samples or features exhibit a common scale. If two zero mean stochastic variables are scaled both with the same factors their magnitude and variance will increase together. That is, as the magnitudes of one variable increase so will the magnitude and the variance of the other variable. This results in a broadening of the histogram of one variable as one increases the value of the conditioning variable — resulting in a "bow-tie" shaped conditional density.

## 4   Relationship to spherical invariant random process

A closely related class of signals to those in Equation 1 is the so-called Spherical Invariant Random Process (SIRP). If the signals are short time Gaussian and the powers vary slowly the class of signals described are approximately SIRPs. Despite the restriction to Gaussian distributed $z$ SIRPs have been shown to be a good model for a range of stochastic processes with very different higher-order properties, depending on the scale distributions $p_s(s)$. They have been used in a variety of signal processing applications [6]. Band-limited speech, in particular, has been shown to be well described by SIRPs [3]. If $z$ is multidimensional, such as a window of samples in a time series or a multi-dimensional feature vector, one talks about Spherically Invariant Random Vectors SIRVs. Natural images have been modeled by what in essence is closely related to SIRVs — a infinite mixture of zero mean Gaussian features [11]. Similar models have also been used for financial time series [2].

The fundamental property of SIRPs is that the joint distribution of a SIRP is entirely defined by a univariate characteristic function $C_x(u)$ and the covariance $\Sigma$ of neighboring samples [6]. They are directly related to our scale-non-stationarity model through a theorem by Kingman and Yao which states that any SIRP is

equivalent to a zero mean Gaussian process $z(t)$ with an independent stochastic scale $s$. Furthermore the univariate characteristic function $C_x(u)$ specifies $p_s(s)$ and the 1D marginal $p_x(x)$ and visa versa [6]. From the characteristic function $C_x(u)$ and the covariance $\Sigma$ one can also construct all higher dimensional joint densities. This leads to the following relation between the marginal densities of various orders [3],

$$p_n(\mathbf{x}) = \pi^{-n/2} f_n(\mathbf{x}^T \Sigma^{-1} \mathbf{x}), \quad \text{with } \mathbf{x} \in \mathbb{R}^n, \text{ and } \Sigma = \langle \mathbf{x}\mathbf{x}^T \rangle, \tag{5}$$

$$f_{n+2}(s) = -\frac{d}{ds} f_n(s), \qquad f_{2m}(s) = \pi^{-1/2} \int_{-\infty}^{\infty} f_{2m+1}(s + y^2)\, dy \tag{6}$$

In particular these relations allow us to compute the joint density $p_2(x(t), x(t+1))$ from an empirically estimated marginal density $p_1(x(t))$ and the covariance of $x(t)$ and $x(t+1)$. Comparing the resulting 2D joint density to the observed joint density allows to us verify the assumption that the data is sampled from a SIRP. In so doing we can more firmly assert that the observed two dimensional joint histograms can in fact be explained as a Gaussian process with a non-stationary scale.

If we use zero mean Gaussian mixtures, $p_1(x) = \sum_{i=1}^{K} m_i \exp(-x^2/\sigma_i^2)$, as the 1D model distribution the resulting 2D joint distribution is simply $p_n(x) = \sum_{i=1}^{K} m_i \exp(-\mathbf{x}^T \Sigma^{-1} \mathbf{x}/\sigma_i^2)$. If the model density is given by a Meijer-G function, as suggested in [3] with $p_1(x) = \frac{\lambda}{\Gamma^2(\lambda)} G_{02}^{20}(\lambda^2 x^2 | \lambda - 0.5, \lambda - 0.5)$, then the 2D joint is $p_2(\mathbf{x}) = \frac{\lambda^2}{\sqrt{\pi}\Gamma^2(\lambda)} G_{13}^{30}(\lambda^2 \mathbf{x}^T \Sigma^{-1} \mathbf{x} | -0.5; 0, \lambda, \lambda)$. In both cases it is assumed that the data is normalized to unit variance.

Brehm has used this approach to demonstrate that band-limited speech is well described by a SIRP [3]. In addition, we show here that the same is true for the image features and stock market data presented above. The model conditional densities shown in Figure 2 correspond well with the empirical conditional histograms. In particular they exhibit the characteristic bow-tie structure. We emphasize that these model 2D joint densities have been obtained only from the 1D marginal of Figure 1 and the covariance of neighboring samples.

The deviations of the observed and model 2D joint distributions are likely due to variable covariance itself, that is, not only does the overall scale or power vary with time, but the components of the covariance matrix vary independently of each other. For example in speech the covariance of neighboring samples is well known to change considerably over time. Nevertheless, the surprising result is that a simple scale non-stationarity model can reproduce the higher-order statistical properties in a variety of natural signals.

## 5  Spectro-temporal linear basis for speech

As an example of the utility of this non-stationarity assumption, we analyze the statistical properties of the *powers* of a *single source*, in particular for speech signals. Motivated by the auditory spectro-temporal receptive field reported in [5] and work on receptive fields and independent components we are interested to find a linear basis of independent components in a spectro-temporal window of speech signals. In [9, 8] we show that one can use second order statistic to uniquely recover sources from a mixture provided that the mix is linear and the sources are non-stationary. One can do so by finding a basis that guarantees uncorrelated signals at multiple time intervals (multiple decorrelation algorithm (MDA)). Our present model argues that features of natural signals such as the powers in different frequency bands can be assumed non-stationary, while powers of independent signals are known to add

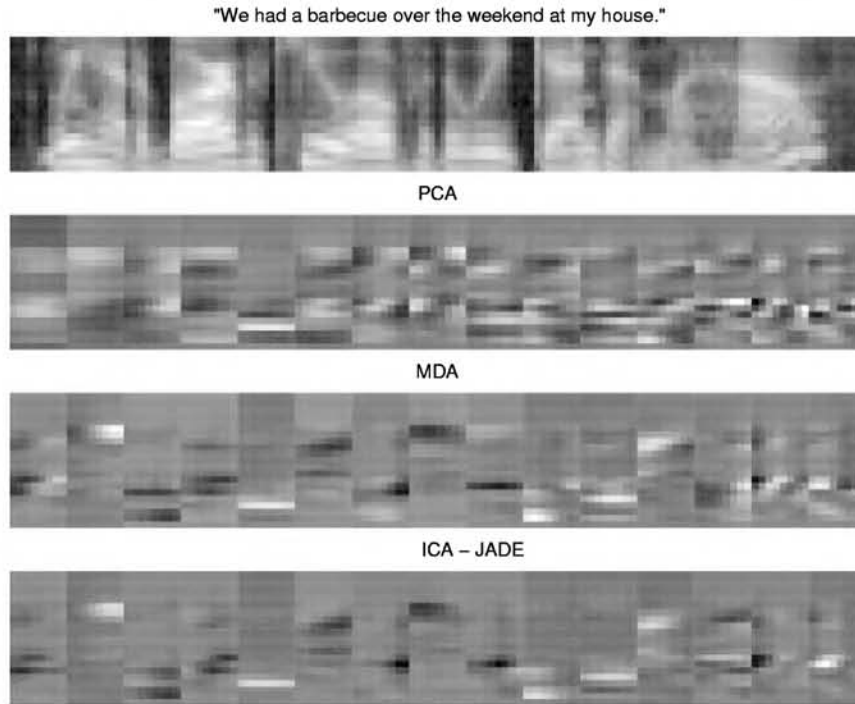

"We had a barbecue over the weekend at my house."

PCA

MDA

ICA – JADE

Figure 3: Spectro-temporal representation of speech. One pixel in the horizontal direction corresponds to 16 ms. In the vertical direction 21 Bark scale power bands are displayed. The upper diagram shows the log-powers for a 2.5 s segment of the 200 s recording used to compute the different linear bases. The three lower diagrams show three sets of 15 linear basis components for 21x8 spectro-temporal segments of the speech powers. The sets correspond to PCA, MDA, and ICA respectively. Note that these are not log-powers, hence the smaller contribution of the high frequencies as compared to the log-power plot on top.

linearly. We should be able therefore to identify with second order methods the same linear components as with independent component algorithms where high-order statistical assumptions are invoked.

We compute the powers in 21 frequency bands on a Bark scale for short consecutive time intervals. We choose to find a basis for a segment of 21 bands and 8 neighboring time slices corresponding to 128 ms of signal between 0 and 4 kHz. We used half overlapping windows of 256 samples such that for a 8 kHz signal neighboring time slices are 16 ms apart. A set of 7808 such spectro-temporal segments were sampled from 200 s of the same speech data presented previously. Figure 3 shows the results obtained for a subspace of 15 components. One can see that the components obtained with MDA are quite similar to the result of ICA and differ considerably from the principal components. From this we conclude that speech powers can in fact be thought of as a linear combination of non-stationary independent components. In general, the point we wish to make is to demonstrate the strength of second-order methods when the assumptions of non-stationarity, independence, and linear superposition are met.

# 6 Conclusion

We have presented evidence that several high-order statistical properties of natural signals can be explained by a simple scale non-stationary model. For four types of natural signals, we have shown that a scale non-stationary model will reproduce the high-kurtosis behavior of the marginal densities. Furthermore, for the case of scale non-stationary with Gaussian density (SIRP), we have shown that we can reproduce the variance dependency seen in conditional histograms of the joint density directly from the empirical marginal densities. This leads to the conclusion that a scale non-stationary model (e.g. SIRP) is a good model for these natural signals. We have shown that one can exploit the assumptions of this model to compute a linear basis for natural signals without having to invoke higher order statistically techniques. Though we do not claim that all higher-order properties or all natural signals can be explained by a scale non-stationary model, it is remarkable that such a simple model can account for a variety of the higher-order phenomena and for a variety of signal types.

## Footnotes

[1] Throughout this paper we will refer to signals that are sampled in time. Note that all the arguments apply equally well to a spatial rather than temporal sampling, that is, images rather than time series.

# References

[1] E.M.L. Beale and C.L. Mallows. Scale mixing of symmetric distributions with zero means. *Annals of Mathematical Statitics*, 30:1145–1151, 1959.

[2] T. P. Bollerslev, R. F. Engle, and D. B. Nelson. Arch models. In R. F. Engle and D. L. McFadden, editors, *Handbook of Econometrics*, volume IV. North-Holland, 1994.

[3] Helmut Brehm and Walter Stammler. Description and generation of spherically invariant speech-model signals. *Signal Processing*, 12:119–141, 1987.

[4] Phil Brodatz. *Textures: A Photographic Album for Artists and Designers.* Dover, 1999.

[5] R. deCharms, Christopher and M. Merzenich, Miachael. Characteristic neuros in the primary auditory cortex of the awake primate using reverse correlation. In M. Jordan, M. Kearns, and S. Solla, editors, *Advances in Neural Information Processing Systems 10*, pages 124–130, 1998.

[6] Joel Goldman. Detection in the presence of spherically symmetric random vectors. *IEEE Transactions on Information Theory*, 22(1):52–59, January 1976.

[7] M.G. Kendal and A. Stuart. *The Advanced Theory of Statistics.* Charles Griffin & Company Limited, London, 1969.

[8] L. Parra and C. Spence. Convolutive blind source separation of non-stationary sources. *IEEE Trans. on Speech and Audio Processing*, pages 320–327, May 2000.

[9] Lucas Parra and Clay Spence. Separation of non-stationary sources. In Stephen Roberts and Richard Everson, editors, *Independent Components Analysis: Principles and Practice.* Cambridge University Press, 2001.

[10] Akaysha Tang, Barak Pearlmutter, Dan Phung, and Scott Carter. Independent components of magnetoencephalography. *Neural Computation*, submitted.

[11] Martin J. Wainwright and Eero P. Simoncelli. Scale mixtures of Gaussians and the statistics of natural images. In S. A. Solla, T.K. Leen, and K.-R. Müller, editors, *Advances in Neural Information Processing Systems 12*, pages 855–861, Cambridge, MA, 2000. MIT Press.
